# A Bayesian Model for Simultaneous Image Clustering, Annotation and Object Segmentation

**Lan  Du, Lu  Ren, [1]David B. Dunson and Lawrence  Carin**
Department of Electrical and Computer Engineering
[1]Statistics Department
Duke University
Durham, NC 27708-0291, USA
`{ld53, lr, lcarin}`@ee.duke.edu,dunson@stats.duke.edu

## Abstract

A non-parametric Bayesian model is proposed for processing multiple images. The analysis employs image features and, when present, the words associated with accompanying annotations. The model clusters the images into classes, and each image is segmented into a set of objects, also allowing the opportunity to assign a word to each object (localized labeling). Each object is assumed to be represented as a heterogeneous mix of components, with this realized via mixture models linking image features to object types. The number of image classes, number of object types, and the characteristics of the object-feature mixture models are inferred nonparametrically. To constitute spatially contiguous objects, a new logistic stick-breaking process is developed. Inference is performed efficiently via variational Bayesian analysis, with example results presented on two image databases.

## 1   Introduction

There has recently been much interest in developing statistical models for analyzing and organizing images, based on image features and, when available, auxiliary information, such as words (*e.g.*, annotations). Three important aspects of this problem are: ($i$) sorting multiple images into scene-level classes, ($ii$) image annotation, and ($iii$) segmenting and labeling localized objects within images. Probabilistic topic models, originally developed for text analysis [8, 12], have been adapted and extended successfully for many image-understanding problems [3, 6, 9–11, 16, 23, 24]. Moreover, recent work has also used the Dirichlet process (DP) [5] or similar non-parametric priors to enhance the topic-model structure [2, 20, 26]. Using such statistical models, researchers [2, 3, 6, 10, 16, 20, 23, 24, 26] have addressed two or all three of the objectives simultaneously within a single setting. Such unified formalisms have realized marked improvements in overall algorithm performance. A relatively complete summary of the literature may be found in [16, 23], where the advantages of the approaches in [16, 23] are described relative to previous related approaches [3, 6, 10, 11, 18, 24, 27]. The work in [16, 23] is based on the correspondence LDA (Corr-LDA) model [6]. The approach in [23] integrates the Corr-LDA model and the supervised LDA (sLDA) model [7] into a single framework. Although good classification performance was achieved using this approach, the model is employed in a supervised manner, utilizing scene-labeled images for scene classification. A class label variable is introduced in [16] to cluster all images in an unsupervised manner, and a switching variable to address noisy annotations. Nevertheless, to improve performance, in [16] some images are required for supervised learning, based on the segmented and labeled objects obtained via the method proposed in [10], with these used to initialize the algorithm.

The research reported here seeks to build upon and extend recent research on unified image-analysis models. Specifically, motivated by [16, 23], we develop a novel non-parametric Bayesian model

that simultaneously addresses all three objectives discussed above. The four main contributions of this paper are:

• Each object in an image is represented as a mixture of image-feature model parameters, accounting for the heterogeneous character of individual objects. This framework captures the idea that a particular object may be composed as an aggregation of distinct parts. By contrast, each object is only associated with one image-feature component/atom in the Corr-LDA-like models [6, 16, 23].

• Multiple images are processed jointly; all, none or a subset of the images may be annotated. The model infers the linkage between image-feature parameters and object types, with this linkage used to yield localized labeling of objects within all images. The unsupervised framework is executed *without* the need for a human to constitute training data.

• A novel logistic stick-breaking process (LSBP) is proposed, imposing the belief that proximate portions of an image are more likely to reside within the same segment (object). This spatially constrained prior yields contiguous objects with sharp boundaries, and via the aforementioned mixture models the segmented objects may be composed of heterogeneous building blocks.

• The proposed model is nonparametric, based on use of stick-breaking constructions [13], which can be easily implemented by fast variational Bayesian (VB) inference [14]. The number of image classes, number of object types, number of image-feature mixture components per object, and the linkage between words and image model parameters are inferred nonparametrically.

## 2 The Hierarchical Generative Model

### 2.1 Bag of image features

We jointly process data from $M$ images, and each image is assumed to come from an associated class type (*e.g.*, city scene, beach scene, office scene, etc.). The class type associated with image $m$ is denoted by $z_m \in \{1, \ldots, I\}$, and it is drawn from the mixture model

$$z_m \sim \sum_{i=1}^{I} u_i \delta_i \quad , \quad \boldsymbol{u} \sim \text{Stick}_I(\alpha_u) \tag{1}$$

where $\text{Stick}_I(\alpha_u)$ is a stick-breaking process [13] that is truncated to $I$ sticks, with hyper-parameter $\alpha_u > 0$. The symbol $\delta_i$ represents a unit measure at the integer $i$, and the parameter $u_i$ denotes the probability that image type $i$ will be observed across the $M$ images.

The observed data are image feature vectors, each tied to a local region in the image (for example, associated with an over-segmented portion of the image). The $L_m$ observed image feature vectors associated with image $m$ are $\{\boldsymbol{x}_{ml}\}_{l=1}^{L_m}$, and the $l$th feature vector is assumed drawn $\boldsymbol{x}_{ml} \sim F(\boldsymbol{\theta}_{ml})$. The expression $F(\cdot)$ represents the feature model, and $\boldsymbol{\theta}_{ml}$ represents the model parameters.

Each image is assumed to be composed of a set of latent objects. An indicator variable $\zeta_{ml}$ defines which object type the $l$th feature vector from image $m$ is associated with, and it is drawn

$$\zeta_{ml} \sim \sum_{k=1}^{K} w_{z_m k} \delta_k \quad , \quad \boldsymbol{w}_i \sim \text{Stick}_K(\alpha_w) \tag{2}$$

where index $k$ corresponds to the $k$th type of object that may reside within an image. The vector $\boldsymbol{w}_i$ defines the probability that each of the $K$ object types will occur, conditioned on the image type $i \in \{1, \ldots, I\}$; the $k$th component of $\boldsymbol{w}_{z_m}$, $w_{z_m k}$, denotes the probability of observing object type $k$ in image $m$, when image $m$ was drawn from class $z_m \in \{1, \ldots, I\}$.

The image class $z_m$ and corresponding objects $\{\zeta_{ml}\}_{l=1}^{L_m}$ associated with image $m$ are latent variables. The generative process for the observed data, $\{\boldsymbol{x}_{ml}\}_{l=1}^{L_m}$, is manifested via mixture models with respect to model parameter $\boldsymbol{\theta}$. Specifically, a separate such mixture model is manifested for each of the $K$ object types, motivated by the idea that each object will in general be composed of a different set of image-feature building blocks. The mixture model for object type $k \in \{1, \ldots, K\}$ is represented as

$$G_k = \sum_{j=1}^{J} h_{kj} \delta_{\boldsymbol{\theta}_j^*} \quad , \quad \boldsymbol{h}_k \sim \text{Stick}_J(\alpha_h) \quad , \quad \boldsymbol{\theta}_j^* \sim H \tag{3}$$

where $H$ is a base measure, usually selected to be conjugate to $F(\cdot)$.

### 2.2 Bag of clustered image features

While the model described above is straightforward to understand, it has been found to be ineffective. This is because each of the $\zeta_{ml}$ is drawn *i.i.d.* from $\sum_{k=1}^{K} w_{z_m k} \delta_k$, and therefore there is

nothing in the model that encourages the image features, $\boldsymbol{x}_{ml}$ and $\boldsymbol{x}_{ml'}$, which are associated with the same image-feature atom $\boldsymbol{\theta}_j^*$, to be assigned to the same object $k$.

To address this limitation, we add a clustering step within each of the images; this is similar to the structure of the hierarchical Dirichlet process (HDP) [21]. Specifically, consider the following augmented model:

$$\boldsymbol{x}_{ml} \sim F(\boldsymbol{\theta}_{ml}) \ , \ \boldsymbol{\theta}_{ml} \sim G_{c_{ml}} \ , \ c_{ml} \sim \sum_{t=1}^{T} v_{mt}\delta_{\zeta_{mt}} \ , \ \zeta_{mt} \sim \sum_{k=1}^{K} w_{z_m k}\delta_k \ , \ z_m \sim \sum_{i=1}^{I} u_i\delta_i \quad (4)$$

where $\boldsymbol{v}_m \sim \text{Stick}_T(\alpha_v)$, and $G_k$ is as defined in (3). We make truncation level $T < K$, to encourage a relatively small number of objects in a given image.

### 2.3 Linking words with images

In the above discussion it was assumed that the only observed data are the image feature vectors $\{\boldsymbol{x}_{ml}\}_{l=1}^{L_m}$. However, there are situations for which annotations (words) may be available for at least a subset of the $M$ images. In this setting we assume that we have a $K$-dimensional dictionary of words associated with objects in images, and a word is assigned to each of the objects $k \in \{1, \ldots, K\}$. Of the collection of $M$ images, some may be annotated and some not, and all will be processed simultaneously by the joint model; in so doing, annotations will be inferred for the originally non-annotated images.

For an image for which no annotation is given, the image is assumed generated via (4). When an annotation is available, the words associated with image $m$ are represented as a vector $\boldsymbol{y}_m = [y_{m1}, \cdots, y_{mK}]^\text{T}$, where $y_{mk}$ denotes the number of times word $k$ is present in the annotation to image $m$ (typically $y_{mk}$ will either be one or zero), and $\boldsymbol{y}_m$ is assumed drawn from a multinomial distribution associated with a parameter $\boldsymbol{\varphi}_m$: $\boldsymbol{y}_m \sim \text{Mult}(\boldsymbol{\varphi}_m)$. If image $m$ is in class $z_m$, then we simply set

$$\boldsymbol{y}_m \sim \text{Mult}(\boldsymbol{w}_{z_m}) \ , \ \ \boldsymbol{w}_i \sim \text{Stick}_K(\alpha_w) \quad (5)$$

Namely, $\boldsymbol{\varphi}_m = \boldsymbol{w}_{z_m}$, recalling that $\boldsymbol{w}_i$ defines the probability of observing each object type for image class $i$. When a dictionary of $K$ words is available, we generally use $\boldsymbol{w}_i \sim \text{Dir}(\alpha_w/K, \ldots, \alpha_w/K)$, consistent with LDA [8].

## 3 Encouraging Spatially Contiguous Objects

### 3.1 Logistic stick-breaking process (LSBP)

In (5), note that once the image class $z_m$ is drawn for image $m$, the order/location of the $\boldsymbol{x}_{ml}$ within the image may be interchanged, and nothing in the generative process will change. This is because the indicator variable $c_{ml}$, which defines the object class associated with feature vector $l$ in image $m$, is drawn *i.i.d.* $c_{ml} \sim \sum_{t=1}^{T} v_{mt}\delta_{\zeta_{mt}}$. It is therefore desirable to impose that if two feature vectors are proximate within the image, they are likely to be associated with the same object.

With each feature vector $\boldsymbol{x}_{ml}$ there is an associated spatial location, which we denote $\boldsymbol{s}_{ml}$ (this is a two-dimensional vector). We wish to draw

$$c_{ml} \sim \sum_{t=1}^{T} v_{mt}(\boldsymbol{s}_{ml})\delta_{\zeta_{mt}} \ , \ \ \ \zeta_{mt} \sim \sum_{k=1}^{K} w_{z_m k}\delta_k \quad (6)$$

where the cluster probabilities $v_{mt}(\boldsymbol{s}_{ml})$ are now a function of position $\boldsymbol{s}_{ml}$ (the $\zeta_{mt} \in \{1, \ldots, K\}$ correspond to object types). The challenge, therefore, becomes development of a means of constructing $v_{mt}(\boldsymbol{s})$ to encourage nearby feature vectors to come from the same object type. Toward this goal, let $\sigma[g_{mt}(\boldsymbol{s})]$ represent a logistic link function, which is a function of $\boldsymbol{s}$. For $t = 1, \ldots, T-1$ we impose

$$v_{mt}(\boldsymbol{s}) = \sigma[g_{mt}(\boldsymbol{s})] \prod_{\tau=1}^{t-1} \{1 - \sigma[g_{m\tau}(\boldsymbol{s})]\} \quad (7)$$

where $v_{mT}(\boldsymbol{s}) = 1 - \sum_{t=1}^{T-1} v_{mt}(\boldsymbol{s})$. We define $g_{mt}(\boldsymbol{s}) = \sum_{l=1}^{L_m} W_{tl}^{(m)}\mathcal{K}(\boldsymbol{s}, \boldsymbol{s}_{ml}) + W_{t0}^{(m)}$ where $\mathcal{K}(\boldsymbol{s}, \boldsymbol{s}_{ml})$ is a kernel, and here we utilize the radial basis function kernel $\mathcal{K}(\boldsymbol{s}, \boldsymbol{s}_{ml}) = \exp[-\|\boldsymbol{s} - \boldsymbol{s}_{ml}\|_2/\phi_{mt}]$. The parameter kernel width $\phi_{mt}$ plays an important role in dictating the size of segments associated with stick $t$, and therefore these parameters should be *learned* by the data in the analysis. In practice we define a library of discrete kernel widths $\boldsymbol{\phi}^* = \{\phi_d^*\}_{d=1}^D$, and infer each $\phi_{mt}$, placing a uniform prior on the elements of $\boldsymbol{\phi}^*$.

We desire that a given stick $v_{mt}(s)$ has importance (at most) over a localized region, and therefore we impose sparseness priors on parameters $\{W_{tl}^{(m)}\}_{l=0}^{L_m}$. Specifically, $W_{tl}^{(m)} \sim \mathcal{N}(0, (\eta_{tl}^{(m)})^{-1})$, and $\eta_{tl}^{(m)}$ is drawn from a gamma prior, with hyper-parameters set to encourage most $\eta_{tl}^{(m)} \to \infty$. Such a Student-t prior is also applied in [4]. The model described above is termed a logistic stick-breaking process (LSBP). For notational convenience, $c_{ml} \sim \sum_{t=1}^{T} v_{mt}(s_{ml})\delta_{\zeta_{mt}}$ and $\zeta_{mt} \sim \sum_{k=1}^{K} w_{z_m k}\delta_k$ constructed as above is represented as a draw from $\text{LSBP}_T(w_{z_m})$. Figure 1 depicts the detailed generative process of the proposed model with LSBP.

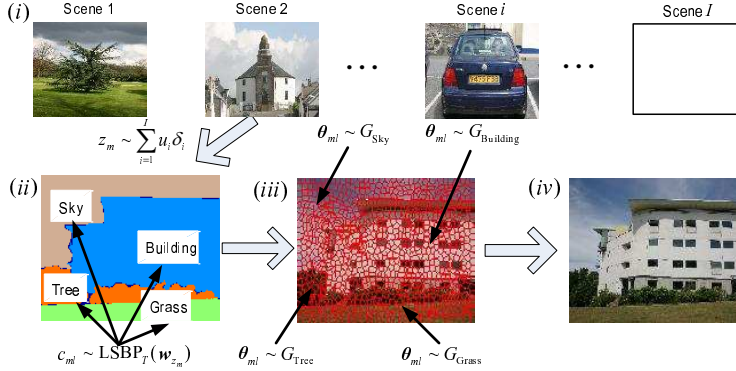

Figure 1: Depiction of the generative process. ($i$) A scene-class indicator $z_m \in \{1, \ldots, I\}$ is drawn to define the image class; ($ii$) conditioned on $z_m$, and using the LSBP, contiguous segmented blocks are constituted, with associated words defined by object indicator $c_{ml} \in \{1, \cdots, K\}$, where $w_i$ defines the probability of observing each object type for image class $i$; ($iii$) conditioned on $c_{ml}$, image-feature atoms are drawn from appropriate mixture models $G_{c_{ml}}$, linked to over-segmented regions within each of the object clusters; ($iv$) the image-feature model parameters are responsible for generating the image features, via the model $F(\theta)$, where $\theta$ is the image-feature parameter.

## 3.2   Discussion of LSBP properties and comparison with KSBP

There are two key components of the LSBP construction: ($i$) sparseness promotion on the $W_{tl}^{(m)}$, and ($ii$) the use of a logistic link function to define spatial stick weights. A particular non-zero $W_{tl}^{(m)}$ is (via the kernel) associated with the $l$th local spatial region, with spatial extent defined by $\phi_{mt}$. If $W_{tl}^{(m)}$ is sufficiently large, the "clipping" property of the logistic link yields a spatially contiguous and extended region over which the $t$th LSBP layer will dominate. Specifically, $c_{ml}^{(t)}$ will likely be the same for data samples located near (defined by $\phi_{mt}$) where a large $W_{tl}^{(m)}$ resides, since in this region $\sigma[g_{mt}(s)] \to 1$. All locations $s$ for which (roughly) $g_{mt}(s) \geq 4$ will have – via the "clipping" manifested via the logistic – nearly the same high probability of being associated with model layer $t$. Sharp segment boundaries are also encouraged by the steep slope of the logistic function.

A related use of spatial information is constituted via the kernel stick-breaking process (KSBP) [2]. With the KSBP, rather than assuming exchangeable data, the $v_{mt}(s)$ in (6) is defined as:

$$v_{mt}(s) = V_{mt}\mathcal{K}(s, \Gamma_{mt}) \prod_{\tau}^{t-1}[1 - V_{mt}\mathcal{K}(s, \Gamma_{m\tau}; \phi)], \quad V_{mt} \sim \text{Beta}(1, \alpha_0) \tag{8}$$

where $\mathcal{K}(s, \Gamma_{mt})$ represents a kernel distance between the feature-vector spatial coordinate $s$ and a local basis location $\Gamma_{mt}$ associated with the $t$th stick. Although such a model also establishes spatial dependence within local regions, the form of the prior has not been found explicit enough to impose smooth segments with sharp boundaries, as demonstrated in [2].

## 4   Using the Proposed Model

### 4.1   Inference

Bayesian inference seeks to estimate the posterior distribution of the latent variables $\Psi$, given the observed data $D$ and hyper-parameters $\Upsilon$. We employ variational Bayesian (VB) [14] inference as a compromise between accuracy and efficiency. This method approximates an intractable joint posterior $p(\Psi|D)$ of all the hidden variables by a product of marginal distributions $q(\Psi) = \prod_f q_f(\Psi_f)$, each over only a single hidden variable $\Psi_f$. The optimal parameterization of $q_f(\Psi_f)$ for each variable is obtained by minimizing the Kullback-Leibler divergence between the variational approximation $q(\Psi)$ and the true joint posterior $p(\Psi)$.

### 4.2 Processing images with no words given

If one is given $M$ images, all non-annotated, then the model may be employed on the data $\{x_{ml}\}_{l=1}^{L_m}$, for $m = 1, \ldots, M$, from which a posterior distribution is inferred on the image model parameters $\{\theta_j^*\}_{j=1}^J$, and on $\{G_k\}_{k=1}^K$. Note that properties of the image classes and of the objects within images is inferred by processing all $M$ images jointly. By placing all images within the context of each other, the model is able to infer which building blocks (classes and objects) are responsible for all of the data. In this sense the simultaneous processing of multiple images is critical: the learning of properties of objects in one image is aided by the properties being learned for objects in all other images, through the inference of inter-relationships and commonalities.

After the $M$ images are analyzed in the absence of annotations, one may observe example portions of the $M$ images, to infer the link between actual object characteristics within imagery and the associated latent object indicator to which it was assigned. With this linkage made, one may assign words to all or a subset of the $K$ object types. After words are assigned to previously latent object types, the results of the analysis (with no additional processing) may be used to automatically label regions (objects) in *all* of the images. This is manifested because each of the cluster indicators $c_{ml}$ is associated with a latent localized object type (to which a word may now be assigned).

### 4.3 Joint processing of images and annotations

We may consider problems for which a subset of the images are provided with *annotations* (but not the explicit location and segmented-out objects); the words are assumed to reside in a prescribed dictionary of object types. The generation of the annotations (and images) is constituted via the model in (5), with the LSBP employed as discussed. We do not require that all images are annotated (the non-annotated images help learn the properties of the image features, and are therefore useful even if they do not provide information about the words). It is desirable that the same word be annotated for multiple images. The presence of the same word within the annotations of multiple images encourages the model to infer what objects (represented in terms of image features) are common to the associated images, aiding the learning. Hence, the presence of annotations serves as a learning aid (encourages looking for commonalities between particular images, if words are shared in the associated annotations). Further, the annotations associated with images may disambiguate objects that appear similar in image-feature space (because they will have different annotations).

From the above discussion, the model performance will improve as more images are annotated with each word, but presumably this annotation is much easier for the human than requiring one to segment out and localize words within a scene.

## 5 Experimental Results

Experiments are performed on two real-world data sets: subsets of Microsoft Research (MSRC) data ( *http://research.microsoft.com/en-us/projects/objectclassrecognition/* ) and UIUC-Sport data from [15, 16], the latter images originally obtained from the Flickr website and available online ( *http://vision.cs.princeton.edu/lijiali/event_dataset/* ).

For the MSRC dataset, 10 categories of images with manual annotations are selected: "tree", "building", "cow", "face", "car", "sheep", "flower", "sign", "book" and "chair". The number of images in the "cow" class is 45, and in the "sheep" class there are 35; there are 30 images in all other classes. From each category, we randomly choose 10 images, and remove the annotations, treating these as non-annotated images within the analysis (to allow quantification of inferred-annotation quality). Each image is of size $213 \times 320$ or $320 \times 213$. In addition, we remove all words that occur less that 8 times (approximately 1% of all words). There are 14 unique words: "void", "building", "grass", "tree", "cow", "sheep", "sky", "face", "car", "flower", "sign", "book", "chair" and "road". We assume that each word corresponds to a visual object in the image. Regarding the case in which multiple words may refer to the same object, one may use the method mentioned in [16] to group synonyms in the preprocessing phase (not necessary here). The following analysis, in which annotated and non-annotated images are processed jointly, is executed as discussed in Section 4.3.

The UIUC-Sport dataset [15, 16] contains 8 types of sports: "badminton", "bocce", "croquet", "polo", "rock climbing", "rowing", "sailing" and "snowboarding". Here we randomly choose 25 images for each category, and each image is resized to a dimension of $240 \times 320$ or $320 \times 240$. Since the annotations are not available at the cited website, the analysis is initially performed with no words, as discussed in Section 4.2. After performing this analysis, and upon examining the properties of segmented data associated with each (latent) object class on a small subset of the data,

we can infer words associated with some important $G_k$, and then label portions (objects) within each image via the inferred words. This process is different than in [6, 16, 23], in which annotations were employed.

When investigating algorithm performance, we make comparisons to Corr-LDA [6]. Our objectives are related to those in [16, 23], but to the authors' knowledge the associated software is not currently available. The Corr-LDA model [6] is relatively simple, and has been coded ourselves. We also examine our model with the proposed LSBP replaced with with KSBP.

## 5.1 Image preprocessing

Each image is first segmented into 800 "superpixels", which are local, coherent and preserve most of the structure necessary for segmentation at the scale of interest [19]. The software used for over-segmentation is discussed in [17] and is available online (*http://www.cs.sfu.ca/~mori/research/superpixels/*). Each superpixel is represented by both color and texture descriptors, based on the local RGB, hue [25] feature vectors and also the output of maximum response (MR) filter banks [22] (*http://www.robots.ox.ac.uk/~vgg/research/texclass/filters.html*). We discretize these features using a codebook of size 64 (other codebook sizes gave similar performance), and then calculate the distribution [1] for each feature within each superpixel as visual words [3, 6, 10, 11, 20, 23, 24].

Since each superpixel is represented by three visual words, the mixture atoms $\boldsymbol{\theta}_j^*$ are three multinomial distributions $\{\mathrm{Mult}(\boldsymbol{\Theta}_{1j}^*) \bigotimes \mathrm{Mult}(\boldsymbol{\Theta}_{2j}^*) \bigotimes \mathrm{Mult}(\boldsymbol{\Theta}_{3j}^*)\}$ for $j = 1, \cdots, J$. Accordingly, the variational distribution in the VB [14] analysis is $q(\boldsymbol{\theta}_j^*) = \mathrm{Dir}(\boldsymbol{\Theta}_{1j}^*|\tilde{\boldsymbol{\rho}}_{1j}) \bigotimes \mathrm{Dir}(\boldsymbol{\Theta}_{2j}^*|\tilde{\boldsymbol{\rho}}_{2j}) \bigotimes \mathrm{Dir}(\boldsymbol{\Theta}_{3j}^*|\tilde{\boldsymbol{\rho}}_{3j})$.

The center of each superpixel is recorded as the location coordinate $\boldsymbol{s}_{ml}$. The set of discrete kernel widths $\boldsymbol{\phi}^*$ are defined by $30, 35, \cdots, 160$, and a uniform multinomial prior is placed on these parameters (the size of each kernel is inferred, for each of the $T$ LSBP layers, and separately in each of the $M$ images). To save computational resources, rather than centering a kernel at each of the $L_m$ points associated with the superpixels, the kernel spatial centers are placed once every 20 superpixels.

We set truncation levels $I = 20$, $J = 50$ and $T = 10$ (similar results were found for larger truncations). For analysis on UIUC-Sport dataset, $K = 40$. All gamma priors for precision parameters $\alpha_w$, $\alpha_v$ or $\{\eta_{tl}^{(m)}\}_{t=1,l=0,m=1}^{T,L_m,M}$, $\alpha_u$ and $\alpha_h$ are set as $(10^{-6}, 10^{-6})$. All these hyper-parameters and truncation levels have not been optimized or tuned. In the following comparisons, the number of topics is set to be same as the atom number, $J = 50$, and the Dirichlet hyperparameters are set as $(1/J, \ldots, 1/J)^{\mathrm{T}}$ for Corr-LDA model; a gamma prior is also used for the KSBP precision parameter, $\alpha_0$ in (8), also set as $(10^{-6}, 10^{-6})$.

## 5.2 Scene clustering

The proposed model automatically learns a posterior distribution on mixture-weights $\boldsymbol{u}$ and in so doing infers an estimate of the proper number of scene classes. As shown in Figure 2, although we initialized the truncation level to $I = 20$, for the MSRC dataset only the first 10 clusters are selected as being important (the mixture weights for other clusters are very small); recall that "truth" indicated that there were 10 classes. In addition, based on the learned posterior word distribution $\boldsymbol{w}_i$ for each image class $i$, we can further infer which words/objects are probable for each scene class. In Figure 2, we show two example $\boldsymbol{w}_i$ for the MSRC "building" and "cow" classes. Although not shown here for brevity, the analysis on UIUC features correctly inferred the 8 image classes associated with that data (without using annotations). By examining the words and segmented objects extracted with high probability as represented by $\boldsymbol{w}_i$, we may also assign names to each of the 18 image classes across both the MSRC and UIUC data, consistent with the associated class labels provided with the data.

For each image $m \in \{1, \ldots, M\}$ we also have a posterior distribution on the associated class indicator $z_m$. We approximate the membership for each image by assigning it to the mixture with largest probability. This "hard" decision is employed to provide scene-level label for each image (the Bayesian analysis can also yield a "soft" decision in terms of a full posterior distribution). Figure 3 presents the confusion matrices for the proposed model with and without LSBP, on both the MSRC and UIUC datasets. Both forms of the model yield relatively good results, but the average accuracy indicates that the model with LSBP performs better than that without LSBP for both datasets. Note

that the results in Figure 3 for the UIUC-Sport data cannot be directly compared with those in [6, 16], since our experiments were performed on non-annotated images.

Using the concepts discussed in Section 4.2, and employing results from the processed non-annotated UIUC-Sport data, we examined the properties of segmented data associated with each (latent) object type. We inferred the presence of 12 unique objects, and these objects were assigned the following words: "human", "horse", "grass", "sky", "tree", "ground","water", "rock", "court", "boat", "sailboat" and "snow". Using these words, we annotated each image and re-trained our model in the presence of annotations. After doing so, the average accuracies of scene-level clustering are improved to 72.0% and 69.0% with and without LSBP, respectively. The improvement in performance, relative to processing the images without annotations, is attributed to the ability of words to disambiguate distinct objects that have similar properties in image-feature space (*e.g.*, the distinct use of "boat" and "sailboat", which helps distinguish rowing and sailing).

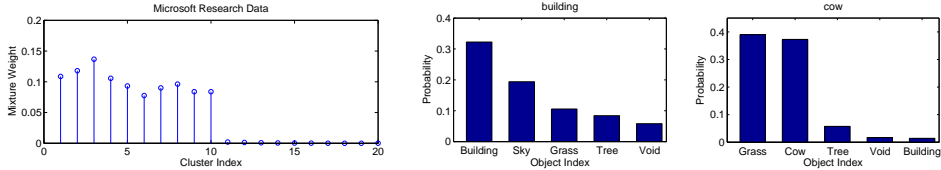

Figure 2: Example inferred latent properties associated with MSRC dataset. Left: Posterior distribution on the mixture-weights $u$, quantifying the probability of scene classes (10 classes are inferred). Middle and Right: Example probability of objects for a given class, $w_i$ (probability of object/words); here we only give the top 5 words for each class.

**without LSBP (MSRC)**

| | tree | building | cow | face | car | sheep | flower | sign | book | chair |
|---|---|---|---|---|---|---|---|---|---|---|
| tree | .83 | .13 | .00 | .03 | .00 | .00 | .00 | .00 | .00 | .00 |
| building | .10 | .80 | .00 | .03 | .00 | .00 | .00 | .07 | .00 | .00 |
| cow | .04 | .02 | .87 | .00 | .00 | .07 | .00 | .00 | .00 | .00 |
| face | .03 | .10 | .00 | .73 | .00 | .00 | .07 | .07 | .00 | .00 |
| car | .03 | .10 | .00 | .00 | .87 | .00 | .00 | .00 | .00 | .00 |
| sheep | .03 | .03 | .09 | .00 | .00 | .86 | .00 | .00 | .00 | .00 |
| flower | .00 | .07 | .00 | .03 | .00 | .00 | .83 | .07 | .00 | .00 |
| sign | .03 | .03 | .00 | .00 | .00 | .00 | .10 | .80 | .03 | .00 |
| book | .00 | .00 | .00 | .03 | .00 | .00 | .00 | .13 | .83 | .00 |
| chair | .10 | .03 | .00 | .00 | .00 | .00 | .00 | .00 | .00 | .87 |

**with LSBP (MSRC)**

| | tree | building | cow | face | car | sheep | flower | sign | book | chair |
|---|---|---|---|---|---|---|---|---|---|---|
| tree | .87 | .10 | .00 | .03 | .00 | .00 | .00 | .00 | .00 | .00 |
| building | .13 | .83 | .00 | .03 | .00 | .00 | .00 | .00 | .00 | .00 |
| cow | .04 | .00 | .89 | .00 | .00 | .07 | .00 | .00 | .00 | .00 |
| face | .03 | .07 | .00 | .80 | .00 | .00 | .07 | .03 | .00 | .00 |
| car | .00 | .17 | .00 | .00 | .83 | .00 | .00 | .00 | .00 | .00 |
| sheep | .00 | .00 | .11 | .00 | .00 | .89 | .00 | .00 | .00 | .00 |
| flower | .00 | .00 | .00 | .10 | .00 | .00 | .87 | .03 | .00 | .00 |
| sign | .00 | .07 | .00 | .00 | .00 | .00 | .03 | .87 | .03 | .00 |
| book | .00 | .00 | .00 | .03 | .00 | .00 | .00 | .10 | .87 | .00 |
| chair | .07 | .03 | .00 | .00 | .00 | .00 | .00 | .00 | .00 | .90 |

**without LSBP (UIUC-Sport)**

| | badmi. | bocce | croquet | polo | rockc | sailing | rowing | snowb. |
|---|---|---|---|---|---|---|---|---|
| badmi. | .76 | .00 | .08 | .04 | .00 | .04 | .04 | .04 |
| bocce | .08 | .44 | .24 | .04 | .04 | .08 | .00 | .08 |
| croquet | .04 | .08 | .72 | .08 | .04 | .04 | .00 | .00 |
| polo | .04 | .04 | .12 | .64 | .04 | .04 | .04 | .04 |
| rockc | .00 | .04 | .04 | .00 | .76 | .04 | .04 | .08 |
| sailing | .04 | .04 | .04 | .04 | .00 | .44 | .32 | .08 |
| rowing | .04 | .04 | .04 | .04 | .04 | .28 | .44 | .08 |
| snowb. | .04 | .08 | .04 | .04 | .08 | .04 | .04 | .64 |

**with LSBP (UIUC-Sport)**

| | badmi. | bocce | croquet | polo | rockc | sailing | rowing | snowb. |
|---|---|---|---|---|---|---|---|---|
| badmi. | .76 | .04 | .04 | .04 | .00 | .04 | .04 | .04 |
| bocce | .04 | .48 | .24 | .04 | .04 | .04 | .04 | .08 |
| croquet | .04 | .08 | .72 | .08 | .04 | .04 | .00 | .00 |
| polo | .04 | .04 | .12 | .64 | .04 | .04 | .04 | .04 |
| rockc | .00 | .08 | .04 | .00 | .76 | .04 | .04 | .08 |
| sailing | .04 | .04 | .00 | .04 | .00 | .52 | .28 | .08 |
| rowing | .04 | .04 | .04 | .04 | .04 | .24 | .52 | .04 |
| snowb. | .04 | .08 | .00 | .04 | .08 | .04 | .04 | .68 |

Figure 3: Comparisons using confusion matrices for all images in each dataset (all of the annotated and non-annotated images in MSRC; all the non-annotated images in UIUC-Sport). The left two results are for MSRC, and the right two for UIUC-Sport. In each pair, the result is without LSBP, and the right is with LSBP. Average performance, left to right: 82.90%, 86.80%, 60.50% and 63.50%.

## 5.3   Image annotation

The proposed model infers a posterior distribution for the indicator variables $c_{ml}$ (defining the object/word for super-pixel $l$ in image $m$). Similar to the "hard" image-class assignment discussed above, a "hard" segmentation is employed here to provide object labels for each super-pixel. For the MSRC images for which annotations were held out, we evaluate whether the words associated with objects in a given image were given in the associated annotation (thus, our annotation is defined by the words we have assigned to objects in an image).

Table 1: Comparison of precision and recall values for annotation and segmentation with Corr-LDA [6], our model without LSBP (Simp. Model) and the extended models with KSBP (Ext. with KSBP) and LSBP (Ext. with LSBP) on MSRC datasets. To evaluate annotation performance, the results are just calculated based on non-annotated images; while for segmentation, the results are based on all images.

| | Annotation | | | | | | | | | Segmentation | | | | | | | | | | | |
|---|---|---|---|---|---|---|---|---|---|---|---|---|---|---|---|---|---|---|---|---|---|
| | Corr-LDA | | | Simp. Model | | | Ext. with LSBP | | | Corr-LDA | | | Simp. Model | | | Ext. with KSBP | | | Ext. with LSBP | | |
| Object | Prec | Rec | F | Prec | Rec | F | Prec | Rec | F | Prec | Rec | F | Prec | Rec | F | Prec | Rec | F | Prec | Rec | F |
| car | .18 | .60 | .28 | .70 | .70 | **.70** | .70 | .70 | **.70** | .13 | .08 | .10 | .49 | .38 | .43 | .56 | .50 | .53 | .61 | .58 | **.60** |
| tree | .30 | .50 | .38 | .50 | .60 | .55 | .55 | .60 | **.57** | .06 | .03 | .04 | .43 | .38 | .40 | .48 | .44 | .46 | .51 | .48 | **.50** |
| sheep | .17 | .60 | .27 | .70 | .70 | .70 | .70 | .70 | **.70** | .02 | .02 | .02 | .53 | .63 | .58 | .57 | .63 | .60 | .60 | .62 | **.61** |
| sky | .38 | .65 | .48 | .66 | .60 | .63 | .68 | .60 | **.64** | .39 | .29 | .33 | .40 | .51 | .45 | .49 | .54 | .51 | .55 | .55 | **.55** |
| chair | .14 | .60 | .22 | .70 | .70 | **.70** | .70 | .70 | **.70** | .13 | .16 | .15 | .57 | .55 | .56 | .58 | .55 | **.57** | .59 | .55 | **.57** |
| **Mean** | .23 | .63 | .32 | .65 | .63 | .64 | .67 | .65 | **.65** | .17 | .18 | .16 | .49 | .51 | .50 | .53 | .53 | .53 | .56 | .54 | **.54** |

We use precision-recall and F-measures [16, 23] to quantitatively evaluate the annotation performance. The left part of Table 1 lists detailed annotation results for five objects, as well as the overall scores from all objects classes for the MSRC data. Our annotation results consistently and significantly outperform Corr-LDA, especially for the precision values.

## 5.4 Object segmentation

Figure 4 shows some detailed object-segmentation results of Corr-LDA and the proposed model (with and without LSBP). We observe that our models generally yield visibly better segmentation relative to Corr-LDA. For example, for complicated objects the Corr-LDA segmentation results are very sensitive to the feature variance, and an object is generally segmented into many small, detailed parts. By contrast, due to the imposed mixture structure on each object, our models cluster small parts into one aggregate object. Furthermore, LSBP encourages local contiguous regions to be grouped in the same segment, and therefore it is less sensitive to localized variability. In addition, compared with results shown in [2], which also used the MSRC dataset, one may observe KSBP cannot do as well as LSBP in maintaining spatial contiguity, as discussed in Section 3.2. Due to space limitations, detailed example comparison between LSBP and KSBP will be shown elsewhere in a longer report; the quantitative comparison in Table 1 further demonstrate the advantages of LSBP over KSBP.

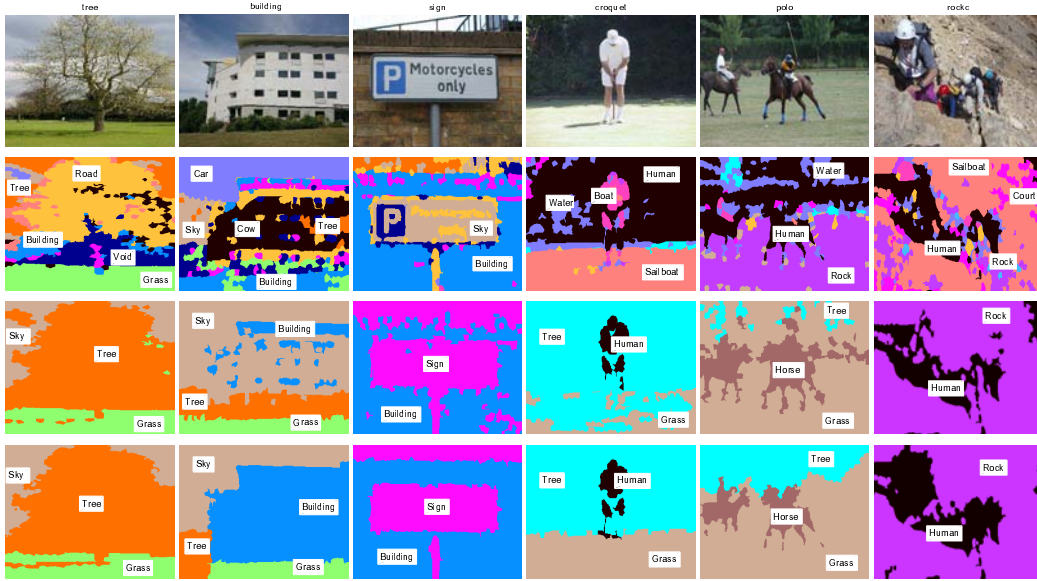

Figure 4: Example segmentation and labeling results. First row: original images; second row: Corr-LDA [6]; third row: proposed model without LSBP; fourth row: proposed model with LSBP. Columns 1-3 from MSRC dataset; Columns 4-6 from UIUC-Sport dataset. The name of original images are inferred by scene-level classification via our model. The UIUC-Sport results are based on the words inferred by our model.

The MSRC database provides manually defined segmentations, to which we quantitatively compare. The right part of Table 1 compares results of the proposed model with Corr-LDA. As indicated in Table 1, the proposed model (with and without LSBP) significantly outperforms Corr-LDA for all objects. Moreover, due to imposed spatial contiguity, the models with KSBP and LSBP are better than without.

The experiments have been performed in non-optimized software written in Matlab, on a Pentium PC with 1.73 GHz CPU and 4G RAM. One VB run of our model with LSBP, for 70 VB iterations, required nearly 7 hours for 320 images from MSRC dataset. Typically 50 VB iterations are required to achieve convergence. The UIUC-Sport data required comparable CPU time. It typically took less than half the CPU time for our model without LSBP on a same dataset. All results are based on a single VB run, with random initialization.

## 6   Conclusions

A nonparametric Bayesian model has been developed for clustering $M$ images into classes; the images are represented as a aggregation of distinct localized objects, to which words may be assigned. To infer the relationships between image objects and words (labels), we only need to make the association between inferred model parameters and words. This may be done as a post-processing step if no words are provided, and it may done *in situ* if all or a subset of the $M$ images are annotated. Spatially contiguous objects are realized via a new logistic stick-breaking process. Quantitative model performance is highly competitive relative to competing approaches, with relatively fast inference realized via variational Bayesian analysis. The authors acknowledge partial support from ARO, AFOSR, DOE, NGA and ONR.

# References

[1] T. Ahonen and M. Pietikäinen. Image description using joint distribution of filter bank responses. *Pattern Recogntion Letters*, 30:368–376, 2009.

[2] Q. An, C. Wang, I. Shterev, E. Wang, L. Carin, and D. B. Dunson. Hierarchical kernel stick-breaking process for multi-task image analysis. In *ICML*, 2008.

[3] K. Barnard, P. Duygulu, N. de Freitas, D. Forsyth, D. M. Blei, and M. I. Jordan. Matching words and pictures. *JMLR*, 3:1107–1135, 2003.

[4] C. M. Bishop and M. E. Tipping. Variational relevance vector machines. In *UAI*, 2000.

[5] D. Blackwell and J. B. MacQueen. Ferguson distributions via Polya urn schemes. *Ann. Statist.*, 1(2):353–355, 1973.

[6] D. M. Blei and M. Jordan. Modeling annotated data. In *SIGIR*, 2003.

[7] D. M. Blei and J. D. McAuliffe. Supervised topic model. In *NIPS*, 2007.

[8] D. M. Blei, A. Ng, and M. I. Jordan. Latent Dirichlet allocation. *JMLR*, 3:993–1022, 2003.

[9] A. Bosch, A. Zisserman, and X. Munoz. Scene classification via plsa. In *ECCV*, 2006.

[10] L. Cao and L. Fei-Fei. Spatially coherent latent topic model for concurrent segmentation and classification of objects and scenes. In *ICCV*, 2007.

[11] L. Fei-Fei and P. Perona. A Bayesian hieratchical model for learning natural scence categories. In *CVPR*, 2005.

[12] T. Hofmann. Unsupervised learning by probabilistic latent semantic analysis. *Mach. Learn.*, 42(1-2):177–196, 2001.

[13] H. Ishwaran and L. F. James. Gibbs sampling methods for stick-breaking priors. *JASA*, 96(453):161–173, 2001.

[14] M. I. Jordan, Z. Ghahramani, T. S. Jaakkola, and L. Saul. An introduction to variational methods for graphical models. *Mach. Learn.*, 37(2):183–233, 1999.

[15] J. Li and L. Fei-Fei. What, where and who? classfying events by scene and object recognition. In *ICCV*, 2007.

[16] J. Li, R. Socher, and L. Fei-Fei. Towards total scene understaning: classification, annotation and segmentation in an automatic framework. In *CVPR*, 2009.

[17] G. Mori. Guiding model search using segmentation. In *ICCV*, 2005.

[18] A. Rabinovich, A. Vedaldi, C. Galleguillos, and E. Wiewiora. Objects in context. In *ICCV*, 2007.

[19] X. Ren and J. Malik. Learning a classification model foe segmentation. In *ICCV*, 2003.

[20] E. B. Sudderth and M. I. Jordan. Shared segementation of natural scenes using dependent pitman-yor processes. In *NIPS*, 2008.

[21] Y. Teh, M. Jordan, M. Beal, and D. Blei. Hierarchical Dirichlet processes. *JASA*, 101:1566–1582, 2005.

[22] M. Varma and A. Zisserman. Classifying images of materials: Achieving viewpoint and illumination independence. In *ECCV*, 2002.

[23] C. Wang, D. M. Blei, and L. Fei-Fei. Simultaneous image classification and annotation. In *CVPR*, 2009.

[24] X. Wang and E. Grimson. Spatial latent dirichlet allocation. In *NIPS*, 2007.

[25] J. V. D. Weijer and C. Schmid. Coloring local feature extraction. In *ECCV*, 2006.

[26] O. Yakhnenko and V. Honavar. Multi-modal hierarchical Dirichlet process model for predicting image annotation and image-object label correspondence. In *SIAM SDM*, 2009.

[27] Z.-H. Zhou and M.-L. Zhang. Mutlti-instance multi-label learning with application to scene classification. In *NIPS*, 2006.

